# Semi-Supervised Support Vector Machines

**Kristin P. Bennett**
Department of Mathematical Sciences
Rensselaer Polytechnic Institute
Troy, NY 12180    bennek@rpi.edu

**Ayhan Demiriz**
Department of Decision Sciences and Engineering Systems
Rensselaer Polytechnic Institute
Troy, NY 12180  demira@rpi.edu

## Abstract

We introduce a semi-supervised support vector machine ($S^3VM$) method. Given a training set of labeled data and a working set of unlabeled data, $S^3VM$ constructs a support vector machine using both the training and working sets. We use $S^3VM$ to solve the transduction problem using overall risk minimization (ORM) posed by Vapnik. The transduction problem is to estimate the value of a classification function at the given points in the working set. This contrasts with the standard inductive learning problem of estimating the classification function at all possible values and then using the fixed function to deduce the classes of the working set data. We propose a general $S^3VM$ model that minimizes both the misclassification error and the function capacity based on all the available data. We show how the $S^3VM$ model for 1-norm linear support vector machines can be converted to a mixed-integer program and then solved exactly using integer programming. Results of $S^3VM$ and the standard 1-norm support vector machine approach are compared on ten data sets. Our computational results support the statistical learning theory results showing that incorporating working data improves generalization when insufficient training information is available. In every case, $S^3VM$ either improved or showed no significant difference in generalization compared to the traditional approach.

# 1 INTRODUCTION

In this work we propose a method for semi-supervised support vector machines (S$^3$VM). S$^3$VM are constructed using a mixture of labeled data (the training set) and unlabeled data (the working set). The objective is to assign class labels to the working set such that the "best" support vector machine (SVM) is constructed. If the working set is empty the method becomes the standard SVM approach to classification [20, 9, 8]. If the training set is empty, then the method becomes a form of unsupervised learning. *Semi-supervised* learning occurs when both training and working sets are nonempty. Semi-supervised learning for problems with small training sets and large working sets is a form of semi-supervised clustering. There are successful semi-supervised algorithms for k-means and fuzzy c-means clustering [4, 18]. Clustering is a potential application for S$^3$VM as well. When the training set is large relative to the working set, S$^3$VM can be viewed as a method for solving the *transduction* problem according to the principle of *overall risk minimization* (ORM) posed by Vapnik at the NIPS 1998 SVM Workshop and in [19, Chapter 10]. S$^3$VM for ORM is the focus of this paper.

In classification, the transduction problem is to estimate the class of each given point in the unlabeled working set. The usual support vector machine (SVM) approach estimates the entire classification function using the principle of *statistical risk minimization* (SRM). In transduction, one estimates the classification function at points within the working set using information from both the training and working set data. Theoretically, if there is adequate training data to estimate the function satisfactorily, then SRM will be sufficient. We would expect transduction to yield no significant improvement over SRM alone. If, however, there is inadequate training data, then ORM may improve generalization on the working set. Intuitively, we would expect ORM to yield improvements when the training sets are small or when there is a significant deviation between the training and working set subsamples of the total population. Indeed, the theoretical results in [19] support these hypotheses.

In Section 2, we briefly review the standard SVM model for structural risk minimization. According to the principles of structural risk minimization, SVM minimize both the empirical misclassification rate and the capacity of the classification function [19, 20] using the training data. The capacity of the function is determined by margin of separation between the two classes based on the training set. ORM also minimizes the both the empirical misclassification rate and the function capacity. But the capacity of the function is determined using both the training and working sets. In Section 3, we show how SVM can be extended to the semi-supervised case and how mixed integer programming can be used practically to solve the resulting problem. We compare support vector machines constructed by structural risk minimization and overall risk minimization computationally on ten problems in Section 4. Our computational results support past theoretical results that improved generalization can be obtained by incorporating working set information during training when there is a deviation between the working set and training set sample distributions. In three of ten real-world problems the semi-supervised approach, S$^3$VM, achieved a significant increase in generalization. In no case did S$^3$VM ever obtain a significant decrease in generalization. We conclude with a discussion of more general S$^3$VM algorithms.

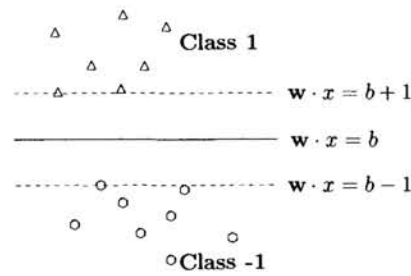

Figure 1: Optimal plane maximizes margin.

## 2   SVM using Structural Risk Minimization

The basic SRM task is to estimate a classification function $f : R^N \rightarrow \{\pm 1\}$ using input-output training data from two classes

$$(\mathbf{x}_1, y_1), \ldots, (\mathbf{x}_\ell, y_\ell) \in R^n \times \{\pm 1\}. \tag{1}$$

The function $f$ should correctly classify unseen examples $(\mathbf{x}, y)$, i.e. $f(\mathbf{x}) = y$ if $(\mathbf{x}, y)$ is generated from the same underlying probability distribution as the training data. In this work we limit discussion to linear classification functions. We will discuss extensions to the nonlinear case in Section 5. If the points are linearly separable, then there exist an $n$-vector $\mathbf{w}$ and scalar $b$ such that

$$\begin{aligned} \mathbf{w} \cdot x_i - b &\geq 1 \quad \textit{if } y_i = 1, \textit{ and} \\ \mathbf{w} \cdot x_i - b &\leq -1 \quad \textit{if } y_i = -1, \; i = 1, \ldots, \ell \end{aligned} \tag{2}$$

or equivalently

$$y_i[\mathbf{w} \cdot x_i - b] \geq 1, \; i = 1, \ldots, \ell. \tag{3}$$

The "optimal" separating plane, $\mathbf{w} \cdot x = b$, is the one which is furthest from the closest points in the two classes. Geometrically this is equivalent to maximizing the separation margin or distance between the two parallel planes $\mathbf{w} \cdot x = b + 1$ and $\mathbf{w} \cdot x = b - 1$ (see Figure 1.)

The "margin of separation" in Euclidean distance is $2/\|\mathbf{w}\|_2$ where $\|\mathbf{w}\|_2 = \sum_{i=1}^n \mathbf{w}_i^2$ is the 2-norm. To maximize the margin, we minimize $\|\mathbf{w}\|_2/2$ subject to the constraints (3). According to structural risk minimization, for a fixed empirical misclassification rate, larger margins should lead to better generalization and prevent overfitting in high-dimensional attribute spaces. The classifier is called a support vector machine because the solution depends only on the points (called support vectors) located on the two supporting planes $\mathbf{w} \cdot x = b - 1$ and $\mathbf{w} \cdot x = b + 1$.

In general the classes will not be separable, so the generalized optimal plane (GOP) problem (4) [9, 20] is used. A slack term $\eta_i$ is added for each point such that if the point is misclassified, $\eta_i \geq 1$. The final GOP formulation is:

$$\begin{aligned} \min_{\mathbf{w}, b, \eta} \quad & C \sum_{i=1}^{\ell} \eta_i + \frac{1}{2} \|\mathbf{w}\|^2 \\ s.t. \quad & y_i[\mathbf{w} \cdot x_i - b] + \eta_i \geq 1 \\ & \eta_i \geq 0, \quad i = 1, \ldots, \ell \end{aligned} \tag{4}$$

where $C > 0$ is a fixed penalty parameter. The capacity control provided by the margin maximization is imperative to achieve good generalization [21, 19].

The Robust Linear Programming (RLP) approach to SVM is identical to GOP except the margin term is changed from the 2-norm $\|\mathbf{w}\|_2$ to the 1-norm, $\|\mathbf{w}\|_1 =$

$\sum_{j=1}^{n} |w_j|$. The problem becomes the following robust linear program (RLP) [2, 7, 1]:

$$\min_{\mathbf{w},b,s,\eta} \quad C\sum_{i=1}^{\ell}\eta_i + \sum_{j=1}^{n}s_j$$
$$s.t. \quad y_i[\mathbf{w}\cdot x_i - b] + \eta_i \geq 1$$
$$\eta_i \geq 0, \quad i = 1,\ldots,\ell \tag{5}$$
$$-s_j <= w_j <= s_j, \quad j = 1,\ldots,n.$$

The RLP formulation is a useful variation of SVM with some nice characteristics. The 1-norm weight reduction still provides capacity control. The results in [13] can be used to show that minimizing $\|\mathbf{w}\|_1$ corresponds to maximizing the separation margin using the infinity norm. Statistical learning theory could potentially be extended to incorporate alternative norms. One major benefit of RLP over GOP is dimensionality reduction. Both RLP and GOP minimize the magnitude of the weights $\mathbf{w}$. But RLP forces more of the weights to be 0 due to the properties of the 1-norm. Another benefit of RLP over GOP is that it can be solved using linear programming instead of quadratic programming. Both approaches can be extended to handle nonlinear discrimination using kernel functions [8, 12]. Empirical comparisons of the approaches have not found any significant difference in generalization between the formulations [5, 7, 3, 12].

## 3   Semi-supervised support vector machines

To formulate the $S^3VM$ , we start with either SVM formulation, (4) or (5), and then add two constraints for each point in the working set. One constraint calculates the misclassification error as if the point were in class 1 and the other constraint calculates the misclassification error as if the point were in class $-1$. The objective function calculates the minimum of the two possible misclassification errors. The final class of the points corresponds to the one that results in the smallest error. Specifically we define the semi-supervised support vector machine problem ($S^3VM$) as:

$$\min_{\mathbf{w},b,\eta,\xi,z} \quad C\left[\sum_{i=1}^{\ell}\eta_i + \sum_{j=\ell+1}^{\ell+k}min(\xi_j,z_j)\right] + \| \mathbf{w} \|$$
$$subject\ to \quad y_i(\mathbf{w}\cdot x_i + b) + \eta_i \geq 1 \quad \eta_i \geq 0 \quad i = 1,\ldots,\ell \tag{6}$$
$$\mathbf{w}\cdot x_j - b + \xi_j \geq 1 \quad \xi_j \geq 0 \quad j = \ell+1,\ldots,\ell+k$$
$$-(\mathbf{w}\cdot x_j - b) + z_j \geq 1 \quad z_j \geq 0$$

where $C > 0$ is a fixed misclassification penalty.

Integer programming can be used to solve this problem. The basic idea is to add a 0 or 1 decision variable, $d_j$, for each point $\mathbf{x}_j$ in the working set. This variable indicates the class of the point. If $d_j = 1$ then the point is in class 1 and if $d_j = 0$ then the point is in class $-1$. This results in the following mixed integer program:

$$\min_{\mathbf{w},b,\eta,\xi,z,d} \quad C\left[\sum_{i=1}^{\ell}\eta_i + \sum_{j=\ell+1}^{\ell+k}(\xi_j + z_j)\right] + \| \mathbf{w} \|$$
$$subject\ to \quad y_i(\mathbf{w}\cdot x_i - b) + \eta_i \geq 1 \quad \eta_i \geq 0 \quad i = 1,\ldots,\ell \tag{7}$$
$$\mathbf{w}\cdot x_j - b + \xi_j + M(1-d_j) \geq 1 \quad \xi_j \geq 0 \quad j = \ell+1,\ldots,\ell+k$$
$$-(\mathbf{w}\cdot x_j - b) + z_j + Md_j \geq 1 \quad z_j \geq 0 \quad d_j = \{0,1\}$$

The constant $M > 0$ is chosen sufficiently large such that if $d_j = 0$ then $\xi_j = 0$ is feasible for any optimal $\mathbf{w}$ and $b$. Likewise if $d_j = 1$ then $z_j = 0$. A globally optimal

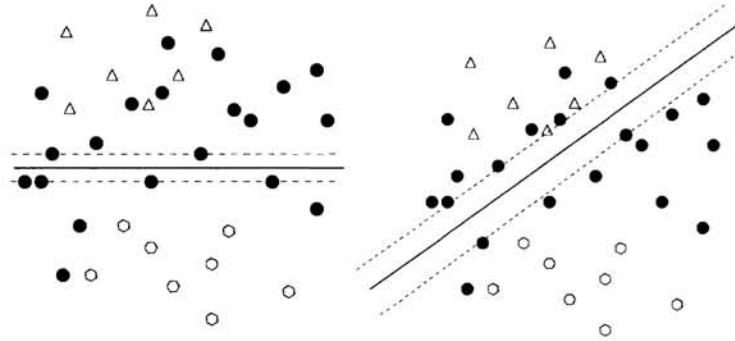

Figure 2: Left = solution found by RLP; Right = solution found by $S^3$VM

solution to this problem can be found using CPLEX or other commercial mixed integer programming codes [10] provided computer resources are sufficient for the problem size. Using the mathematical programming modeling language AMPL [11], we were able to express the problem in thirty lines of code plus a data file and solve it using CPLEX.

## 4  $S^3$VM and Overall Risk Minimization

An integer $S^3$VM can be used to solve the Overall Risk Minimization problem. Consider the simple problem given in Figure 20 of [19]. Using RLP alone on the training data results in the separation shown in Figure 1. Figure 2 illustrates what happens when working set data is added. The training set points are shown as transparent triangles and hexagons. The working set points are shown as filled circles. The left picture in Figure 2 shows the solution found by RLP. Note that when the working set points are added, the resulting separation has very a small margin. The right picture shows the $S^3$VM solution constructed using the unlabeled working set. Note that a much larger and clearer separation margin is found. These computational solutions are identical to those presented in [19].

We also tested $S^3$VM on ten real-world data sets (eight from [14] and the bright and dim galaxy sets from [15]). There have been many algorithms applied successfully to these problems without incorporate working set information. Thus it was not clear *a priori* that $S^3$VM would improve generalization on these data sets. For the data sets where no improvement is possible, we would like transduction using ORM to not degrade the performance of the induction via SRM approach. For each data set, we performed 10-fold cross-validation. For the three starred data sets, our integer programming solver failed due to excessive branching required within the CPLEX algorithm. On those data sets we randomly extracted 50 point working sets for each trial. The same C parameter was used for each data set in both the RLP and $S^3$VM problems[1]. In all ten problems, $S^3$VM never performed significantly worse than RLP. In three of the problems, $S^3$VM performed significantly better. So ORM did not hurt generalization and in some cases it helped significantly. We would expect this based on ORM theory. The generalization bounds for ORM depend on the difference between the training and working sets. If there is little difference, we would not expect any improvement using ORM.

| Data Set | Dim | Points | CV-size | RLP | $S^3$VM | p-value |
|---|---|---|---|---|---|---|
| Bright | 14 | 2462 | 50* | 0.02 | 0.018 | 0.343 |
| Cancer | 9 | 699 | 70 | 0.036 | 0.034 | 0.591 |
| Cancer(Prognostic) | 30 | 569 | 57 | 0.035 | 0.033 | 0.678 |
| Dim | 14 | 4192 | 50* | 0.064 | 0.054 | 0.096 |
| Heart | 13 | 297 | 30 | 0.173 | 0.160 | 0.104 |
| Housing | 13 | 506 | 51 | 0.155 | 0.151 | 0.590 |
| Ionosphere | 34 | 351 | 35 | 0.109 | 0.106 | 0.59 |
| Musk | 166 | 476 | 48 | 0.173 | 0.173 | 0.999 |
| Pima | 8 | 769 | 50* | 0.220 | 0.222 | 0.678 |
| Sonar | 60 | 208 | 21 | 0.281 | 0.219 | 0.045 |

## 5  Conclusion

We introduced a semi-supervised SVM model. $S^3$VM constructs a support vector machine using all the available data from both the training and working sets. We show how the $S^3$VM model for 1-norm linear support vector machines can be converted to a mixed-integer program. One great advantage of solving $S^3$VM using integer programming is that the globally optimal solution can be found using packages such as CPLEX. Using the integer $S^3$VM we performed an empirical investigation of transduction using overall risk minimization, a problem posed by Vapnik. Our results support the statistical learning theory results that incorporating working data improves generalization when insufficient training information is available. In every case, $S^3$VM either improved or showed no significant difference in generalization compared to the usual structural risk minimization approach. Our empirical results combined with the theoretical results in [19], indicate that transduction via ORM constitutes a very promising research direction.

Many research questions remain. Since transduction via overall risk minimization is not always be better than the basic induction via structural risk minimization, can we identify *a priori* problems likely to benefit from transduction? The best methods of constructing $S^3$VM for the 2-norm case and for nonlinear functions are still open questions. Kernel based methods can be incorporated into $S^3$VM. The practical scalability of the approach needs to be explored. We were able to solve moderately-sized problems with on the order of 50 working set points using a general purpose integer programming code. The recent success of special purpose algorithms for support vector machines [16, 17, 6] indicate that such approaches may produce improvement for $S^3$VM as well.

## Footnotes

[1]The formula for C was $C = \frac{(1-\lambda)}{\lambda(\ell+k)}$ with $\lambda = .001$, $\ell$ is the size of training set, and $k$ is the size of the working set. This formula was chosen because it worked well empirically for both methods.

## References

[1] K. P. Bennett and E. J. Bredensteiner. Geometry in learning. In C. Gorini, E. Hart, W. Meyer, and T. Phillips, editors, *Geometry at Work*, Washington, D.C., 1997. Mathematical Association of America. To appear.

[2] K. P. Bennett and O. L. Mangasarian. Robust linear programming discrimination of two linearly inseparable sets. *Optimization Methods and Software*, 1:23–34, 1992.

[3] K. P. Bennett, D. H. Wu, and L. Auslender. On support vector decision trees for database marketing. R.P.I. Math Report No. 98-100, Rensselaer Polytechnic

Institute, Troy, NY, 1998.

[4] A.M. Bensaid, L.O. Hall, J.C. Bezdek, and L.P. Clarke. Partially supervised clustering for image segmentation. *Pattern Recognition*, 29(5):859–871, 199.

[5] P. S. Bradley and O. L. Mangasarian. Feature selection via concave minimization and support vector machines. Technical Report Mathematical Programming Technical Report 98-03, University of Wisconsin-Madison, 1998. To appear in ICML-98.

[6] P. S. Bradley and O. L. Mangasarian. Massive data discrimination via linear support vector machines. Technical Report Mathematical Programming Technical Report 98-05, University of Wisconsin-Madison, 1998. Submitted for publication.

[7] E. J. Bredensteiner and K. P. Bennett. Feature minimization within decision trees. *Computational Optimization and Applications*, 10:110–126, 1997.

[8] C. J. C Burges. A tutorial on support vector machines for pattern recognition. *Data Mining and Knowledge Discovery*, 1998. to appear.

[9] C. Cortes and V. N. Vapnik. Support vector networks. *Machine Learning*, 20:273–297, 1995.

[10] CPLEX Optimization Incorporated, Incline Village, Nevada. *Using the CPLEX Callable Library*, 1994.

[11] R. Fourer, D. Gay, and B. Kernighan. *AMPL A Modeling Language for Mathematical Programming*. Boyd and Frazer, Danvers, Massachusetts, 1993.

[12] T. T. Fries and R. Harrison Fries. Linear programming support vector machines for pattern classification and regression estimation: and the sr algorithm. Research report 706, University of Sheffield, 1998.

[13] O. L. Mangasarian. Parsimonious least norm approximation. Technical Report Mathematical Programming Technical Report 97-03, University of Wisconsin-Madison, 1997. To appear in *Computational Optimization and Applications*.

[14] P.M. Murphy and D.W. Aha. UCI repository of machine learning databases. Department of Information and Computer Science, University of California, Irvine, California, 1992.

[15] S. Odewahn, E. Stockwell, R. Pennington, R Humphreys, and W Zumach. Automated star/galaxy discrimination with neural networks. *Astronomical Journal*, 103(1):318–331, 1992.

[16] E. Osuna, R. Freund, and F. Girosi. Support vector machines: Training and applications. AI Memo 1602, Maassachusets Institute of Technology, 1997.

[17] J. Platt. Sequentional minimal optimization: A fast algorithm for training support vector machines. Technical Report Technical Report 98-14, Microsoft Research, 1998.

[18] M. Vaidyanathan, R.P. Velthuizen, P. Venugopal, L.P. Clarke, and L.O. Hall. Tumor volume measurements using supervised and semi-supervised mri segmentation. In *Artificial Neural Networks in Engineering Conference, ANNIE(1994)*, 1994.

[19] V. N. Vapnik. *Estimation of dependencies based on empirical Data*. Springer, New York, 1982. English translation, Russian version 1979.

[20] V. N. Vapnik. *The Nature of Statistical Learning Theory*. Springer Verlag, New York, 1995.

[21] V. N. Vapnik and A. Ja. Chervonenkis. *Theory of Pattern Recognition*. Nauka, Moscow, 1974. In Russian.